# Automatic Alignment of Local Representations

**Yee Whye Teh and Sam Roweis**
Department of Computer Science, University of Toronto
{ywteh,roweis}@cs.toronto.edu

## Abstract

We present an automatic alignment procedure which maps the disparate internal representations learned by several local dimensionality reduction experts into a single, coherent global coordinate system for the original data space. Our algorithm can be applied to any set of experts, each of which produces a low-dimensional local representation of a high-dimensional input. Unlike recent efforts to coordinate such models by modifying their objective functions [1, 2], our algorithm is invoked after training and applies an efficient eigensolver to post-process the trained models. The post-processing has no local optima and the size of the system it must solve scales with the number of local models rather than the number of original data points, making it more efficient than model-free algorithms such as Isomap [3] or LLE [4].

## 1 Introduction: Local vs. Global Dimensionality Reduction

Beyond density modelling, an important goal of unsupervised learning is to discover compact, informative representations of high-dimensional data. If the data lie on a smooth low dimensional manifold, then an excellent encoding is the coordinates internal to that manifold. The process of determining such coordinates is dimensionality reduction. Linear dimensionality reduction methods such as principal component analysis and factor analysis are easy to train but cannot capture the structure of curved manifolds. Mixtures of these simple unsupervised models [5, 6, 7, 8] have been used to perform *local* dimensionality reduction, and can provide good density models for curved manifolds, but unfortunately such mixtures cannot do dimensionality reduction. They do not describe a single, coherent low-dimensional coordinate system for the data since there is no pressure for the local coordinates of each component to agree.

Roweis et al [1] recently proposed a model which performs global coordination of local coordinate systems in a mixture of factor analyzers (MFA). Their model is trained by maximizing the likelihood of the data, with an additional variational penalty term to encourage the internal coordinates of the factor analyzers to agree. While their model can trade off modelling the data and having consistent local coordinate systems, it requires a user given trade-off parameter, training is quite inefficient (although [2] describes an improved training algorithm for a more constrained model), and it has quite serious local minima problems (methods like LLE [4] or Isomap [3] have to be used for initialization).

In this paper we describe a novel, automatic way to align the hidden representations used by each component of a mixture of dimensionality reducers into a single global representation of the data throughout space. Given an already trained mixture, the alignment is achieved by applying an eigensolver to a matrix constructed from the internal representations of the mixture components. Our method is efficient, simple to implement, and has no local optima in its optimization nor any learning rates or annealing schedules.

## 2   The Locally Linear Coordination Algorithm

Suppose we have a set of data points given by the rows of $\mathcal{X} = [\boldsymbol{x}_1, \boldsymbol{x}_2, \ldots, \boldsymbol{x}_N]^\top$ from a $D$-dimensional space, which we assume are sampled from a $d \ll D$ dimensional manifold. We approximate the manifold coordinates using images $\mathcal{Y} = [\boldsymbol{y}_1, \boldsymbol{y}_2, \ldots, \boldsymbol{y}_N]^\top$ in a $d$ dimensional embedding space. Suppose also that we have already trained, or have been given, a mixture of $K$ local dimensionality reducers. The $k^{\text{th}}$ reducer produces a $d_k$ dimensional internal representation $\boldsymbol{z}_{nk}$ for data point $\boldsymbol{x}_n$ as well as a "responsibility" $r_{nk} \geq 0$ describing how reliable the $k^{\text{th}}$ reducer's representation of $\boldsymbol{x}_n$ is. These satisfy $\sum_k r_{nk} = 1$ and can be obtained, for example, using a gating network in a mixture of experts, or the posterior probabilities in a probabilistic network. Notice that the manifold coordinates and internal representations need not have the same number of dimensions.

Given the data, internal representations, and responsibilities, our algorithm automatically aligns the various hidden representations into a single global coordinate system. Two key ideas motivate the method. First, to use a convex cost function whose unique minimum is attained at the desired global coordinates. Second, to restrict the global coordinates $\boldsymbol{y}_n$ to depend on the data $\boldsymbol{x}_n$ only through the local representations $\boldsymbol{z}_{nk}$ and responsibilities $r_{nk}$, thereby leveraging the structure of the mixture model to regularize and reduce the effective size of the optimization problem. In effect, rather than working with individual data points, we work with large groups of points belonging to particular submodels.

We first parameterize the global coordinates $\boldsymbol{y}_n$ in terms of $r_{nk}$ and $\boldsymbol{z}_{nk}$. Given an input $\boldsymbol{x}_n$, each local model infers its internal coordinates $\boldsymbol{z}_{nk}$ and then applies a *linear* projection $L_k$ and offset $\boldsymbol{l}_k^0$ to these to obtain its guess at the global coordinates. The final global coordinates $\boldsymbol{y}_n$ is obtained by averaging the guesses using the responsibilities as weights:

$$\boldsymbol{y}_n = \sum_k r_{nk}(L_k \boldsymbol{z}_{nk} + \boldsymbol{l}_k^0) = \sum_k \sum_{i=0}^{d_k} r_{nk} z_{nk}^i \boldsymbol{l}_k^i = \sum_j u_{nj} \boldsymbol{l}_j \tag{1}$$

$$\mathcal{Y} = UL \qquad j = j(i,k), \qquad u_{nj} = r_{nk} z_{nk}^i, \qquad \boldsymbol{l}_j = \boldsymbol{l}_k^i \tag{2}$$

where $\boldsymbol{l}_k^i$ is the $i^{\text{th}}$ column of $L_k$, $z_{nk}^i$ is the $i^{\text{th}}$ entry of $\boldsymbol{z}_{nk}$, and $z_{nk}^0 = 1$ is a bias. This process is described in figure 1. To simplify our calculations, we have vectorized the indices $(i,k)$ into a single new index $j(i,k)$, where $j(i,k)$ is an invertible mapping from the domain of $(i,k)$ to $\{1, 2, \ldots, K + \sum_k d_k\}$. For compactness, we will write $j = j(i,k)$. Now define the matrices $U$ and $L$ as $u_{nj} = r_{nk} z_{nk}^i$ and the $j^{\text{th}}$ row of $L$ as $\boldsymbol{l}_j = \boldsymbol{l}_k^i$. Then (1) becomes a system of linear equations (2) with fixed $U$ and unknown parameters $L$.

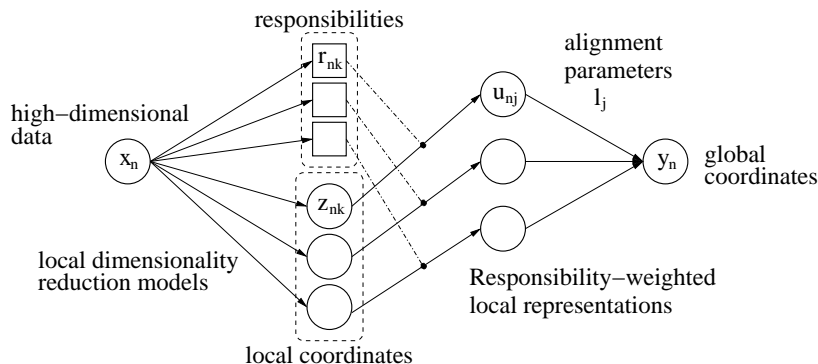

Figure 1: Obtaining global coordinates from data via responsibility-weighted local coordinates.

The key assumption, which we have emphasized by re-expressing $y_n$ above, is that the mapping between the local representations and the global coordinates $y_n$ is linear in each of $z_{nk}$, $r_{nk}$ and the unknown parameters $l_k^i$. Crucially, however, the mapping between the original data $x_n$ and the images $y_n$ is highly non-linear since it depends on the multiplication of responsibilities and internal coordinates which are in turn non-linearly related to the data $x_n$ through the inference procedure of the mixture model.

We now consider determining $L$ according to some given cost function $\mathcal{E}(\mathcal{Y})$. For this we advocate using a convex $\mathcal{E}(\mathcal{Y})$. Notice that since $\mathcal{Y}$ is linear in $L$, $\mathcal{E}(\mathcal{Y}(L))$ is convex in $L$ as well, and there is a unique optimum that can be computed efficiently using a variety of methods. This is still true if we also have feasible convex constraints $\mathcal{C}(\mathcal{Y}) = 0$ on $\mathcal{Y}$. The case where the cost and constraints are both quadratic is particularly appealing since we can use an eigensolver to find the optimal $L$. In particular suppose $Q$ and $R$ are matrices defining the cost and constraints, and let $A = U^\top Q U$ and $B = U^\top R U$. This gives:

$$
\begin{aligned}
\mathcal{E}(\mathcal{Y}) &= \mathrm{Tr}\left[\mathcal{Y}^\top Q \mathcal{Y}\right] & \mathcal{C}(\mathcal{Y}) = 0 &= \mathcal{Y}^\top R \mathcal{Y} - I_d \\
&= \mathrm{Tr}\left[L^\top U^\top Q U L\right] & &= L^\top U^\top R U L - I_d \\
&= \mathrm{Tr}\left[L^\top A L\right] & &= L^\top B L - I_d
\end{aligned}
\tag{3}
$$

where Tr is the trace operator. The matrices $Q$ and $R$ are typically obtained from the original data $\mathcal{X}$ and summarize the essential geometries among them. The solution to the constrained minimization above is given by the $d$ smallest generalized eigenvectors $v$ with $Av = \lambda Bv$. In particular the columns of $L$ are given by these generalized eigenvectors.

Below, we investigate a cost function based on the Locally Linear Embedding (LLE) algorithm of Roweis and Saul [4]. We call the resulting algorithm *Locally Linear Coordination* (LLC). The idea of LLE is to preserve the same locally linear relationships between the original data points $x_n$ and their counterparts $y_n$. We identify for each point $x_n$ its nearest-neighbours $x_m \in \mathcal{N}_n$ and then minimize

$$
\mathcal{E}(\mathcal{X}, W) = \sum_n \left\| x_n - \sum_{m \in \mathcal{N}_n} w_{nm} x_m \right\|^2 = \mathrm{Tr}\left[\mathcal{X}^\top (I - W^\top)(I - W)\mathcal{X}\right]
\tag{4}
$$

with respect to $W$ subject to the constraints $\sum_{m \in \mathcal{N}_n} w_{nm} = 1$. The weights are unique[1] and can be solved for efficiently using constrained least squares (since solving for $w_{nm}$ is decoupled across $n$). The weights summarize the local geometries relating the data points to their neighbours, hence to preserve these relationships among the coordinates $y_n$ we arrange to minimize the same cost

$$
\mathcal{E}(\mathcal{Y}, W) = \mathrm{Tr}\left[\mathcal{Y}^\top (I - W^\top)(I - W)\mathcal{Y}\right]
\tag{5}
$$

but with respect to $\mathcal{Y}$ instead. $\mathcal{E}$ is invariant to translations and rotations of $\mathcal{Y}$, and scales as we scale $\mathcal{Y}$. In order to break these degeneracies we enforce the following constraints:

$$
\frac{1}{N} \sum_n y_n = \frac{1}{N}\vec{1}^\top \mathcal{Y} = 0 \qquad\qquad \frac{1}{N} \sum_n y_n y_n^\top = \frac{1}{N}\mathcal{Y}^\top \mathcal{Y} = I_d
\tag{6}
$$

where $\vec{1}$ is a vector of 1's. For this choice, the cost function and constraints above become:

$$
\mathcal{E}(\mathcal{Y}, W) = \mathrm{Tr}\left[L^\top U^\top (I - W^\top)(I - W) U L\right] = \mathrm{Tr}[L^\top A L]
\tag{7}
$$

$$
\frac{1}{N}\vec{1}^\top U L = 0 \qquad\qquad \frac{1}{N}L^\top U^\top U L = L^\top B L = I_d
\tag{8}
$$

with cost and constraint matrices

$$
A = U^\top (I - W^\top)(I - W) U \qquad B = \frac{1}{N} U^\top U
\tag{9}
$$

[1]In the unusual case where the number of neighbours is larger than the dimensionality of the data $\mathcal{X}$, simple regularization of the norm of the weights once again makes them unique.

As shown previously, the solution to this problem is given by the smallest generalized eigenvectors $\boldsymbol{v}$ with $A\boldsymbol{v} = \lambda B\boldsymbol{v}$. To satisfy $\frac{1}{N}\vec{1}^{\top}UL = 0$, we need to find eigenvectors that are orthogonal to the vector $\boldsymbol{v}_0 = U^{\top}\vec{1}$. Fortunately, $\boldsymbol{v}_0$ *is* the smallest generalized eigenvector, corresponding to an eigenvalue of 0. Hence the solution to the problem is given by the $2^{nd}$ to $(d+1)^{st}$ smallest generalized eigenvectors instead.

---

**LLC Alignment Algorithm:**

- Using data $\mathcal{X}$, compute local linear reconstruction weights $w_{nm}$ using (4).

- Train or receive a pre-trained mixture of local dimensionality reducers. Apply this mixture to $\mathcal{X}$, obtaining a local representation $\boldsymbol{z}_{nk}$ and responsibility $r_{nk}$ for each submodel $k$ and each data point $\boldsymbol{x}_n$.

- Form the matrix $U$ with $u_{nj} = r_{nk}z_{nk}^i$ and calculate $A$ and $B$ from (9).

- Find the eigenvectors corresponding to the smallest $d+1$ eigenvalues of the generalized eigenvalue system $A\boldsymbol{v} = \lambda B\boldsymbol{v}$.

- Let $L$ be a matrix with columns formed by the $2^{nd}$ to $d+1^{st}$ eigenvectors. Return the $j^{th}$ row of $L$ as alignment weight $\boldsymbol{l}_k^i$. Return the global manifold coordinates as $\mathcal{Y} = UL$.

---

Note that the edge size of the matrices $A$ and $B$ whose generalized eigenvectors we seek is $K + \sum_k d_k$ which scales with the number of components and dimensions of the local representations but not with the number of data points $N$. As a result, solving for the alignment weights is much more efficient than the original LLE computation (or those of Isomap) which requires solving an eigenvalue system of edge size $N$. In effect, we have leveraged the mixture of local models to collapse large groups of points together and worked only with those groups rather than the original data points. Notice however that the computation of the weights $W$ still requires determining the neighbours of the original data points, which scales as $O(N^2)$ in the worse case.

Coordination with LLC also yields a mixture of noiseless factor analyzers over the global coordinate space $\boldsymbol{y}$, with the $k^{th}$ factor analyzer having mean $\boldsymbol{l}_k^0$ and factor loading $L_k$. Given any global coordinates $\boldsymbol{y}$, we can infer the responsibilities $r_k$ and the posterior means $\boldsymbol{z}_k$ over the latent space of each factor analyzer. If our original local dimensionality reducers also supports computing $\boldsymbol{x}$ from $r_k$ and $\boldsymbol{z}_k$, we can now infer the high dimensional mean data point $\boldsymbol{x}$ which corresponds to the global coordinates $\boldsymbol{y}$. This allows us to perform operations like visualization and interpolation using the global coordinate system. This is the method we used to infer the images in figures 4 and 5 in the next section.

## 3 Experimental Results using Mixtures of Factor Analyzers

The alignment computation we have described is applicable to any mixture of local dimensionality reducers. In our experiments, we have used the most basic such model: a mixture of factor analyzers (MFA) [8]. The $k^{th}$ factor analyzer in the mixture describes a probabilistic linear mapping from a latent variable $\boldsymbol{z}_k$ to the data $\boldsymbol{x}$ with additive Gaussian noise. The model assumes that the data manifold is locally linear and it is this local structure that is captured by each factor analyzer. The non-linearity in the data manifold is handled by patching multiple factor analyzers together, each handling a locally linear region.

MFAs are trained in an unsupervised way by maximizing the marginal log likelihood of the observed data, and parameter estimation is typically done using the EM algorithm[2].

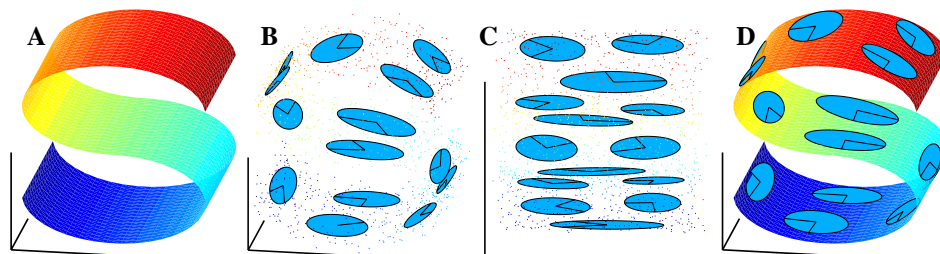

Figure 2: LLC on the S curve (A). There are 14 factor analyzers in the mixture (B), each with 2 latent dimensions. Each disk represents one of them with the two black lines being the factor loadings. After alignment by LLC (C), the curve is successfully unrolled; it is also possible to retroactively align the original data space models (D).

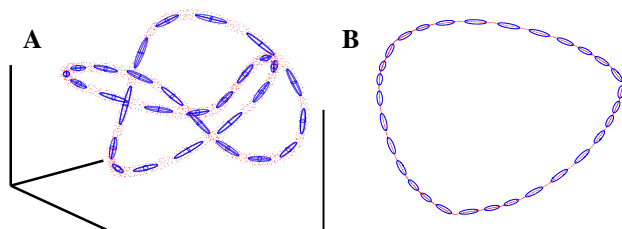

Figure 3: Unknotting the trefoil curve. We generated 6000 noisy points from the curve. Then we fit an MFA with 30 components with 1 latent dimension each (A), but aligned them in a 2D space (B). We used 10 neighbours to reconstruct each data point.

Since there is no constraint relating the various hidden variables $z_k$, a MFA trained only to maximize likelihood cannot learn a global coordinate system for the manifold that is consistent across every factor analyzer. Hence this is a perfect model on which to apply automatic alignment. Naturally, we use the mean of $z_k$ conditioned on the data $x$ (assuming the $k^{\text{th}}$ factor analyzer generated $x$) as the $k^{\text{th}}$ local representation of $x$, while we use the posterior probability that the $k^{\text{th}}$ factor analyzer generated $x$ as the responsibility.

We illustrate LLC on two synthetic toy problems to give some intuition about how it works. The first problem is the S curve given in figure 2(A). An MFA trained on 1200 points sampled uniformly from the manifold with added noise (B) is able to model the linear structure of the curve locally, however the internal coordinates of the factor analyzers are not aligned properly. We applied LLC to the local representations and aligned them in a 2D space (C). When solving for local weights, we used 12 neighbours to reconstruct each data point. We see that LLC has successfully unrolled the S curve onto the 2D space. Further, given the coordinate transforms produced by LLC, we can retroactively align the latent spaces of the MFAs (D). This is done by determining directions in the various latent spaces which get transformed to the *same* direction in the global space.

To emphasize the topological advantages of aligning representations into a space of higher dimensionality than the local coordinates used by each submodel, we also trained a MFA on data sampled from a *trefoil* curve, as shown in figure 3(A). The trefoil is a circle with a knot in 3D. As figure 3(B) shows, LLC connects these models into a ring of local topology faithful to the original data.

We applied LLC to MFAs trained on sets of real images believed to come from a complex manifold with few degrees of freedom. We studied face images of a single person under varying pose and expression changes and handwritten digits from the MNIST database. After training the MFAs, we applied LLC to align the models. The face models were aligned into a 2D space as shown in figure 4. The first dimension appears to describe

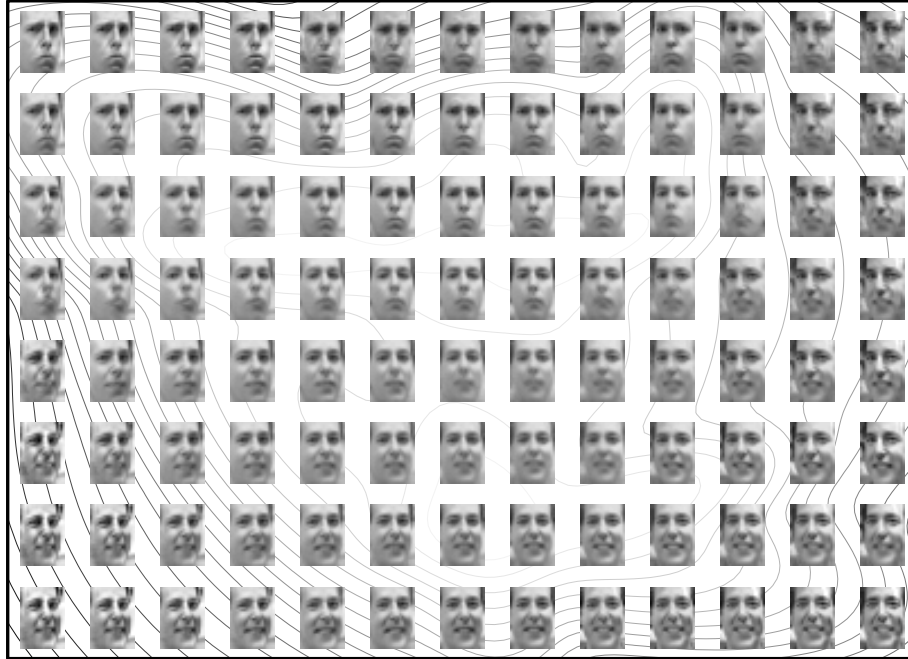

Figure 4: A map of reconstructions of the faces when the global coordinates are specified. Contours describe the likelihood of the coordinates. Note that some reconstructions around the edge of the map are not good because the model is extrapolating from the training images to regions of low likelihood. A MFA with 20 components and 8 latent dimensions each is used. It is trained on 1965 images. The weights $W$ are calculated using 36 neighbours.

changes in pose, while the second describes changes in expression. The digit models were aligned into a 3D space. Figure 5 (top) shows maps of reconstructions of the digits. The first dimension appears to describe the slant of each digit, the second the fatness of each digit, and the third the relative sizes of the upper to lower loops. Figure 5 (bottom) shows how LLC can smoothly interpolate between any two digits. In particular, the first row interpolates between left and right slanting digits, the second between fat and thin digits, the third between thick and thin line strokes, and the fourth between having a larger bottom loop and larger top loop.

## 4  Discussion and Conclusions

Previous work on nonlinear dimensionality reduction has usually emphasized either a parametric approach, which explicitly constructs a (sometimes probabilistic) mapping between the high-dimensional and low-dimensional spaces, or a nonparametric approach which merely finds low-dimensional images corresponding to high-dimensional data points but without probabilistic models or hidden variables. Compared to the global coordination model [1], the closest parametric approach to ours, our algorithm can be understood as *post coordination*, in which the latent spaces are coordinated after they have been fit to data. By decoupling the data fitting and coordination problems we gain efficiency and avoid local optima in the coordination phase. Further, since we are just maximizing likelihood when fitting the original mixture to data, we can use a whole range of known techniques to escape local minima, and improve efficiency in the first phase as well.

On the nonparametric side, our approach can be compared to two recent algorithms, LLE

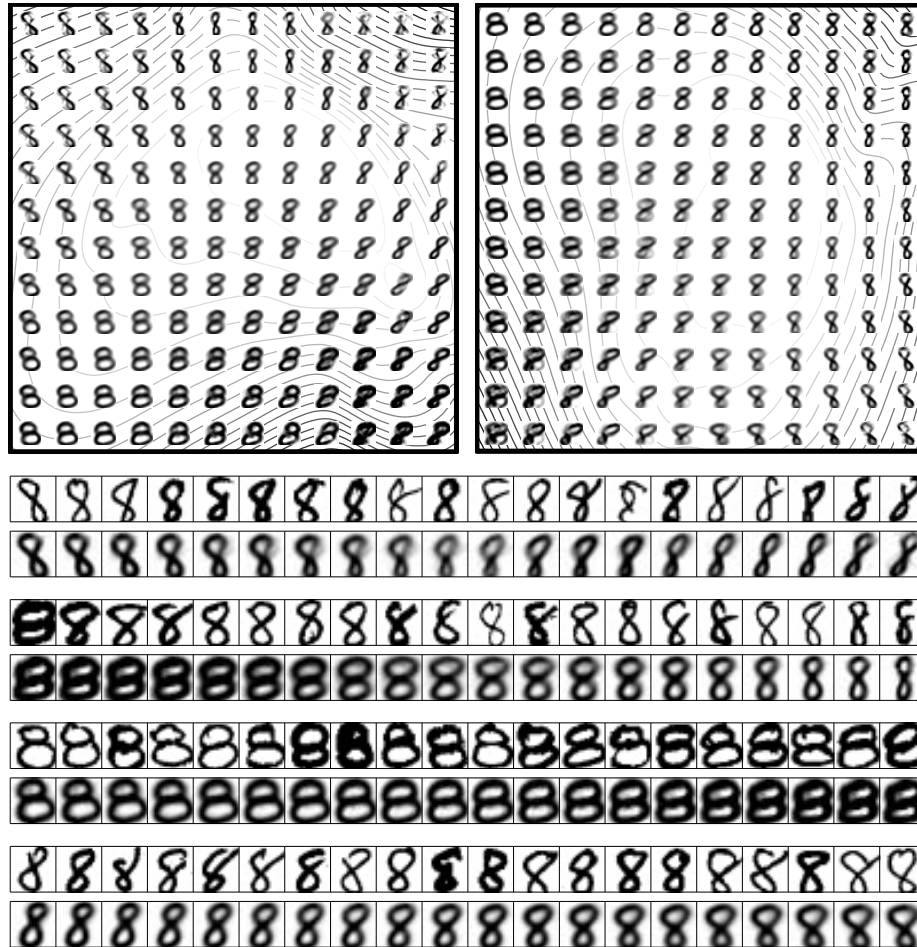

Figure 5: Top: maps of reconstructions of digits when two global coordinates are specified, and the third integrated out. Left: $1^{st}$ and $2^{nd}$ coordinates specified; right: $2^{nd}$ and $3^{rd}$. Bottom: Interpolating between two digits using LLC. In each row, we interpolate between the upper leftmost and rightmost digits. The LLC interpolants are spread out evenly along a line connecting the global coordinates of the two digits. For comparison, we show the 20 training images whose coordinates are closest to the line segment connecting those of the two digits at each side. A MFA with 50 components, each with 6 latent dimensions is used. It is trained on 6000 randomly chosen digits from the combined training and test sets of 8's in MNIST. The weights $W$ were calculated using 36 neighbours.

[4] and Isomap [3]. The cost functions of LLE and Isomap are convex, so they do not suffer from the local minima problems of earlier methods [9, 10], but these methods must solve eigenvalue systems of size equal to the number of data points. (Although in LLE the systems are highly sparse.) Another problem is neither LLE nor Isomap yield a probabilistic model or even a mapping between the data and embedding spaces. Compared to these models (which are run on individual data points) LLC uses as its primitives descriptions of the data provided by the individual local models. This makes the eigenvalue system to be solved much smaller and as a result the computational cost of the coordination phase of LLC is much less than that for LLE or Isomap. (Note that the construction of the eigenvalue system still requires finding nearest neighbours for each point, which is costly.) Furthermore, if each local model describes a complete (probabilistic) mapping from data space

to its latent space, the final coordinated model will also describe a (probabilistic) mapping from the whole data space to the coordinated embedding space.

Our alignment algorithm improves upon local embedding or density models by elevating their status to full global dimensionality reduction algorithms without requiring any modifications to their training procedures or cost functions. For example, using mixtures of factor analyzers (MFAs) as a test case, we show how our alignment method can allow a model previously suited only for density estimation to do complex operations on high dimensional data such as visualization and interpolation.

Brand [11] has recently proposed an approach, similar to ours, that coordinates local parametric models to obtain a globally valid nonlinear embedding function. Like LLC, his "charting" method defines a quadratic cost function and finds the optimal coordination directly and efficiently. However, charting is based on a cost function much closer in spirit to the original global coordination model and it instantiates one local model centred on each training point, so its scaling is the same as that of LLE and Isomap. In principle, Brand's method can be extended to work with fewer local models and our alignment procedure can be applied using the charting cost rather than the LLE cost we have pursued here.

Automatic alignment procedures emphasizes a powerful but often overlooked interpretation of local mixture models. Rather than considering the output of such systems to be a single quantity, such as a density estimate or a expert-weighted regression, it is possible to view them as networks which convert high-dimensional inputs into a *vector of internal coordinates* from each submodel, accompanied by responsibilities. As we have shown, this view can lead to efficient and powerful algorithms which allow separate local models to learn consistent global representations.

## Acknowledgments

We thank Geoffrey Hinton for inspiration and interesting discussions, Brendan Frey and Yann LeCun for sharing their data sets, and the reviewers for helpful comments.

## Footnotes

[2] In our experiments, we initialized the parameters by drawing the means from the global covariance of the data and setting the factors to small random values. We also simplified the factor analyzers to share the same spherical noise covariance $\Phi = \sigma^2 I_N$ although this is not essential to the process.

## References

[1] S. Roweis, L. Saul, and G. E. Hinton. Global coordination of local linear models. In *Advances in Neural Information Processing Systems*, volume 14, 2002.

[2] J. J. Verbeek, N. Vlassis, and B. Kröse. Coordinating principal component analysers. In *Proceedings of the International Conference on Artificial Neural Networks*, 2002.

[3] J. B. Tenenbaum, V. de Silva, and J. C. Langford. A global geometric framework for nonlinear dimensionality reduction. *Science*, 290(5500):2319–2323, December 2000.

[4] S. Roweis and L. Saul. Nonlinear dimensionality reduction by locally linear embedding. *Science*, 290(5500):2323–2326, December 2000.

[5] K. Fukunaga and D. R. Olsen. An algorithm for finding intrinsic dimensionality of data. *IEEE Transactions on Computers*, 20(2):176–193, 1971.

[6] N. Kambhatla and T. K. Leen. Dimension reduction by local principal component analysis. *Neural Computation*, 9:1493–1516, 1997.

[7] M. E. Tipping and C. M. Bishop. Mixtures of probabilistic principal component analysers. *Neural Computation*, 11(2):443–482, 1999.

[8] Z. Ghahramani and G. E. Hinton. The EM algorithm for mixtures of factor analyzers. Technical Report CRG-TR-96-1, University of Toronto, Department of Computer Science, 1996.

[9] T. Kohonen. *Self-organization and Associative Memory*. Springer-Verlag, Berlin, 1988.

[10] C. Bishop, M. Svensen, and C. Williams. GTM: The generative topographic mapping. *Neural Computation*, 10:215–234, 1998.

[11] M. Brand. Charting a manifold. This volume, 2003.
